# Predicting Execution Time of Computer Programs Using Sparse Polynomial Regression

**Ling Huang**
Intel Labs Berkeley
ling.huang@intel.com

**Jinzhu Jia**
UC Berkeley
jzjia@stat.berkeley.edu

**Bin Yu**
UC Berkeley
binyu@stat.berkeley.edu

**Byung-Gon Chun**
Intel Labs Berkeley
byung-gon.chun@intel.com

**Petros Maniatis**
Intel Labs Berkeley
petros.maniatis@intel.com

**Mayur Naik**
Intel Labs Berkeley
mayur.naik@intel.com

## Abstract

Predicting the execution time of computer programs is an important but challenging problem in the community of computer systems. Existing methods require experts to perform detailed analysis of program code in order to construct predictors or select important features. We recently developed a new system to automatically extract a large number of features from program execution on sample inputs, on which prediction models can be constructed *without* expert knowledge. In this paper we study the construction of predictive models for this problem. We propose the SPORE (Sparse POlynomial REgression) methodology to build accurate prediction models of program performance using feature data collected from program execution on sample inputs. Our two SPORE algorithms are able to build relationships between responses (e.g., the execution time of a computer program) and features, and select a few from hundreds of the retrieved features to construct an explicitly sparse and non-linear model to predict the response variable. The compact and explicitly polynomial form of the estimated model could reveal important insights into the computer program (e.g., features and their non-linear combinations that dominate the execution time), enabling a better understanding of the program's behavior. Our evaluation on three widely used computer programs shows that SPORE methods can give accurate prediction with relative error less than 7% by using a moderate number of training data samples. In addition, we compare SPORE algorithms to state-of-the-art sparse regression algorithms, and show that SPORE methods, motivated by real applications, outperform the other methods in terms of both interpretability and prediction accuracy.

## 1 Introduction

Computing systems today are ubiquitous, and range from the very small (e.g., iPods, cellphones, laptops) to the very large (servers, data centers, computational grids). At the heart of such systems are management components that decide how to schedule the execution of different programs over time (e.g., to ensure high system utilization or efficient energy use [11, 15]), how to allocate to each program resources such as memory, storage and networking (e.g., to ensure a long battery life or fair resource allocation), and how to weather anomalies (e.g., flash crowds or attacks [6, 17, 24]).

These management components typically must make guesses about how a program will perform under given hypothetical inputs, so as to decide how best to plan for the future. For example, consider a simple scenario in a data center with two computers, fast computer $A$ and slow computer $B$, and a program waiting to run on a large file $f$ stored in computer $B$. A scheduler is often faced

with the decision of whether to run the program at $B$, potentially taking longer to execute, but avoiding any transmission costs for the file; or moving the file from $B$ to $A$ but potentially executing the program at $A$ much faster. If the scheduler can predict accurately how long the program would take to execute on input $f$ at computer $A$ or $B$, he/she can make an optimal decision, returning results faster, possibly minimizing energy use, etc.

Despite all these opportunities and demands, uses of prediction have been at best unsophisticated in modern computer systems. Existing approaches either create analytical models for the programs based on simplistic assumptions [12], or treat the program as a black box and create a mapping function between certain properties of input data (e.g., file size) and output response [13]. The success of such methods is highly dependent on human experts who are able to select important predictors before a statistical modeling step can take place. Unfortunately, in practice experts may be hard to come by, because programs can get complex quickly beyond the capabilities of a single expert, or because they may be short-lived (e.g., applications from the iPhone app store) and unworthy of the attention of a highly paid expert. Even when an expert is available, program performance is often dependent not on externally visible features such as command-line parameters and input files, but on the internal semantics of the program (e.g., what lines of code are executed).

To address this problem (lack of expert and inherent semantics), we recently developed a new system [7] to automatically extract a large number of features from the intermediate execution steps of a program (e.g., internal variables, loops, and branches) on sample inputs; then prediction models can be built from those features *without* the need for a human expert.

In this paper, we propose two *S*parse *PO*lynomial *RE*gression (SPORE) algorithms that use the automatically extracted features to predict a computer program's performance. They are variants of each other in the way they build the nonlinear terms into the model – *SPORE-LASSO* first selects a small number of features and then entertains a full nonlinear polynomial expansion of order less than a given degree; while *SPORE-FoBa* chooses adaptively a subset of the full expanded terms and hence allows possibly a higher order of polynomials. Our algorithms are in fact new general methods motivated by the computer performance prediction problem. They can learn a relationship between a response (e.g., the execution time of a computer program given an input) and the generated features, and select a few from hundreds of features to construct an explicit polynomial form to predict the response. The compact and explicit polynomial form reveals important insights in the program semantics (e.g., the internal program loop that affects program execution time the most). Our approach is general, flexible and automated, and can adapt the prediction models to specific programs, computer platforms, and even inputs.

We evaluate our algorithms experimentally on three popular computer programs from web search and image processing. We show that our SPORE algorithms can achieve accurate predictions with relative error less than 7% by using a small amount of training data for our application, and that our algorithms outperform existing state-of-the-art sparse regression algorithms in the literature in terms of interpretability and accuracy.

**Related Work.** In prior attempts to predict program execution time, Gupta et al. [13] use a variant of decision trees to predict execution time ranges for database queries. Ganapathi et al. [11] use KCCA to predict time and resource consumption for database queries using statistics on query texts and execution plans. To measure the empirical computational complexity of a program, Trendprof [12] constructs linear or power-law models that predict program execution counts. The drawbacks of such approaches include their need for expert knowledge about the program to identify good features, or their requirement for simple input-size to execution time correlations.

Seshia and Rakhlin [22, 23] propose a game-theoretic estimator of quantitative program properties, such as worst-case execution time, for embedded systems. These properties depend heavily on the target hardware environment in which the program is executed. Modeling the environment manually is tedious and error-prone. As a result, they formulate the problem as a game between their algorithm (player) and the program's environment (adversary), where the player seeks to accurately predict the property of interest while the adversary sets environment states and parameters.

Since expert resource is limited and costly, it is desirable to automatically extract features from program codes. Then machine learning techniques can be used to select the most important features to build a model. In statistical machine learning, feature selection methods under linear regression models such as LASSO have been widely studied in the past decade. Feature selection with

non-linear models has been studied much less, but has recently been attracting attention. The most notable are the SpAM work with theoretical and simulation results [20] and additive and generalized forward regression [18]. Empirical studies with data of these non-linear sparse methods are very few [21]. The drawback of applying the SpAM method in our execution time prediction problem is that SpAM outputs an additive model and cannot use the interaction information between features. But it is well-known that features of computer programs interact to determine the execution time [12]. One non-parametric modification of SpAM to replace the additive model has been proposed [18]. However, the resulting non-parametric models are not easy to interpret and hence are not desirable for our execution time prediction problem. Instead, we propose the SPORE methodology and propose efficient algorithms to train a SPORE model. Our work provides a promising example of interpretable non-linear sparse regression models in solving real data problems.

## 2   Overview of Our System

Our focus in this paper is on algorithms for feature selection and model building. However we first review the problem within which we apply these techniques to provide context [7]. Our goal is to predict how a given program will perform (e.g., its execution time) on a particular input (e.g., input files and command-line parameters). The system consists of four steps.

First, the *feature instrumentation* step analyzes the source code and automatically instruments it to extract values of program features such as *loop counts* (how many times a particular loop has executed), *branch counts* (how many times each branch of a conditional has executed), and *variable values* (the $k$ first values assigned to a numerical variable, for some small $k$ such as 5).

Second, the *profiling* step executes the instrumented program with sample input data to collect values for all created program features and the program's execution times. The time impact of the data collection is minimal.

Third, the *slicing* step analyzes each automatically identified feature to determine the smallest subset of the actual program that can compute the value of that feature, i.e., the *feature slice*. This is the cost of obtaining the value of the feature; if the whole program must execute to compute the value, then the feature is *expensive* and not useful, since we can just measure execution time and we have no need for prediction, whereas if only a little of the program must execute, the feature is cheap and therefore possibly valuable in a predictive model.

Finally, the *modeling* step uses the feature values collected during profiling along with the feature costs computed during slicing to build a predictive model on a small subset of generated features. To obtain a model consisting of low-cost features, we iterate over the modeling and slicing steps, evaluating the cost of selected features and rejecting expensive ones, until only low-cost features are selected to construct the prediction model. At runtime, given a new input, the selected features are computed using the corresponding slices, and the model is used to predict execution time from the feature values.

The above description is minimal by necessity due to space constraints, and omits details on the rationale, such as why we chose the kinds of features we chose or how program slicing works. Though important, those details have no bearing in the results shown in this paper.

At present our system targets a fixed, overprovisioned computation environment without CPU job contention or network bandwidth fluctuations. We therefore assume that execution times observed during training will be consistent with system behavior on-line. Our approach can adapt to modest change in execution environment by retraining on different environments. In our future research, we plan to incorporate candidate features of both hardware (e.g., configurations of CPU, memory, etc) and software environment (e.g., OS, cache policy, etc) for predictive model construction.

## 3   Sparse Polynomial Regression Model

Our basic premise for predictive program analysis is that a *small* but *relevant* set of features may explain the execution time well. In other words, we seek a compact model—an explicit form function of a small number of features—that accurately estimates the execution time of the program.

To make the problem tractable, we constrain our models to the multivariate polynomial family, for at least three reasons. First, a "good program" is usually expected to have polynomial execution time in some (combination of) features. Second, a polynomial model up to certain degree can approximate well many nonlinear models (due to Taylor expansion). Finally, a compact polynomial model can provide an easy-to-understand explanation of what determines the execution time of a program, providing program developers with intuitive feedback and a solid basis for analysis.

For each computer program, our feature instrumentation procedure outputs a data set with $n$ samples as tuples of $\{y_i, \mathbf{x}_i\}_{i=1}^n$, where $y_i \in \mathbb{R}$ denotes the $i^{th}$ observation of execution time, and $\mathbf{x}_i$ denotes the $i^{th}$ observation of the vector of $p$ features. We now review some obvious alternative methods to modeling the relationship between $Y = [y_i]$ and $X = [\mathbf{x}_i]$, point out their drawbacks, and then we proceed to our SPORE methodology.

## 3.1 Sparse Regression and Alternatives

Least square regression is widely used for finding the best-fitting $f(\mathbf{x}, \beta)$ to a given set of responses $y_i$ by minimizing the sum of the squares of the residuals [14]. Regression with subset selection finds for each $k \in \{1, 2, \ldots, m\}$ the feature subset of size $k$ that gives the smallest residual sum of squares. However, it is a combinatorial optimization and is known to be NP-hard [14]. In recent years a number of efficient alternatives based on model regularization have been proposed. Among them, LASSO [25] finds the selected features with coefficients $\hat{\beta}$ given a tuning parameter $\lambda$ as follows:

$$\hat{\beta} = \arg\min_{\beta} \frac{1}{2}\|Y - X\beta\|_2^2 + \lambda \sum_j |\beta_j|. \tag{1}$$

LASSO effectively enforces many $\beta_j$'s to be 0, and selects a small subset of features (indexed by non-zero $\beta_j$'s) to build the model, which is usually sparse and has better prediction accuracy than models created by ordinary least square regression [14] when $p$ is large. Parameter $\lambda$ controls the complexity of the model: as $\lambda$ grows larger, fewer features are selected.

Being a convex optimization problem is an important advantage of the LASSO method since several fast algorithms exist to solve the problem efficiently even with large-scale data sets [9, 10, 16, 19]. Furthermore, LASSO has convenient theoretical and empirical properties. Under suitable assumptions, it can recover the true underlying model [8, 25]. Unfortunately, when predictors are highly correlated, LASSO usually cannot select the true underlying model. The adaptive-LASSO [29] defined below in Equation (2) can overcome this problem

$$\hat{\beta} = \arg\min_{\beta} \frac{1}{2}\|Y - X\beta\|_2^2 + \lambda \sum_j |\frac{\beta_j}{w_j}|, \tag{2}$$

where $w_j$ can be any consistent estimate of $\beta$. Here we choose $w_j$ to be a ridge estimate of $\beta$:

$$w_j = (X^T X + 0.001I)^{-1} X^T Y,$$

where $I$ is the identity matrix.

Technically LASSO can be easily extended to create nonlinear models (e.g., using polynomial basis functions up to degree $d$ of all $p$ features). However, this approach gives us $\binom{p+d}{d}$ terms, which is very large when $p$ is large (on the order of thousands) even for small $d$, making regression computationally expensive. We give two alternatives to fit the sparse polynomial regression model next.

## 3.2 SPORE Methodology and Two Algorithms

Our methodology captures non-linear effects of features—as well as non-linear interactions among features—by using polynomial basis functions over those features (we use terms to denote the polynomial basis functions subsequently). We expand the feature set $\mathbf{x} = \{x_1 \; x_2 \ldots \; x_k\}, k \le p$ to all the terms in the expansion of the degree-$d$ polynomial $(1 + x_1 + \ldots + x_k)^d$, and use the terms to construct a multivariate polynomial function $f(\mathbf{x}, \beta)$ for the regression. We define $expan(X, d)$ as the mapping from the original data matrix $X$ to a new matrix with the polynomial expansion terms up to degree $d$ as the columns. For example, using a degree-2 polynomial with feature set

$\mathbf{x} = \{x_1 \, x_2\}$, we expand out $(1 + x_1 + x_2)^2$ to get terms $1$, $x_1$, $x_2$, $x_1^2$, $x_1 x_2$, $x_2^2$, and use them as basis functions to construct the following function for regression:

$$expan\left([x_1, x_2], 2\right) = [1, [x_1], [x_2], [x_1^2], [x_1 x_2], [x_2^2]],$$
$$f(\mathbf{x}, \beta) = \beta_0 + \beta_1 x_1 + \beta_2 x_2 + \beta_3 x_1^2 + \beta_4 x_1 x_2 + \beta_5 x_2^2.$$

Complete expansion on all $p$ features is not necessary, because many of them have little contribution to the execution time. Motivated by this execution time application, we propose a general methodology called SPORE which is a sparse polynomial regression technique. Next, we develop two algorithms to fit our SPORE methodology.

### 3.2.1 SPORE-LASSO: A Two-Step Method

For a sparse polynomial model with only a few features, if we can preselect a small number of features, applying the LASSO on the polynomial expansion of those preselected features will still be efficient, because we do not have too many polynomial terms. Here is the idea:

**Step 1:** Use the linear LASSO algorithm to select a small number of features and filter out (often many) features that hardly have contributions to the execution time.

**Step 2:** Use the adaptive-LASSO method on the expanded polynomial terms of the selected features (from Step 1) to construct the sparse polynomial model.

Adaptive-LASSO is used in Step 2 because of the collinearity of the expanded polynomial features. Step 2 can be computed efficiently if we only choose a small number of features in Step 1. We present the resulting *SPORE-LASSO* algorithm in Algorithm 1 below.

---
**Algorithm 1** SPORE-LASSO

---
**Input:** response $Y$, feature data $X$, maximum degree $d$, $\lambda_1$, $\lambda_2$
**Output:** Feature index $S$, term index $S_t$, weights $\hat{\beta}$ for $d$-degree polynomial basis.
  1: $\hat{\alpha} = \arg\min_\alpha \frac{1}{2}\|Y - X\alpha\|_2^2 + \lambda_1 \|\alpha\|_1$
  2: $S = \{j : \hat{\alpha}_j \neq 0\}$
  3: $X_{new} = expan(X(S), d)$
  4: $w = (X_{new}^T X_{new} + 0.001I)^{-1} X_{new}^T Y$
  5: $\hat{\beta} = \arg\min_\beta \frac{1}{2}\|Y - X_{new}\beta\|_2^2 + \lambda_2 \sum_j |\frac{\beta_j}{w_j}|$
  6: $S_t = \{j : \hat{\beta}_j \neq 0\}$

---

$X(S)$ in Step 3 of Algorithm 1 is a sub-matrix of $X$ containing only columns from $X$ indexed by $S$. For a new observation with feature vector $X = [x_1, x_2, \ldots, x_p]$, we first get the selected feature vector $X(S)$, then obtain the polynomial terms $X_{new} = expan(X(S), d)$, and finally we compute the prediction: $\hat{Y} = X_{new} \times \hat{\beta}$. Note that the prediction depends on the choice of $\lambda_1, \lambda_2$ and maximum degree $d$. In this paper, we fix $d = 3$. $\lambda_1$ and $\lambda_2$ are chosen by minimizing the Akaike Information Criterion (AIC) on the LASSO solution paths. The AIC is defined as $n \log(\|Y - \hat{Y}\|_2^2) + 2s$, where $\hat{Y}$ is the fitted $Y$ and $s$ is the number of polynomial terms selected in the model. To be precise, for the linear LASSO step (Step 1 of Algorithm 1), a whole solution path with a number of $\lambda_1$ can be obtained using the algorithm in [10]. On the solution path, for each fixed $\lambda_1$, we compute a solution path with varied $\lambda_2$ for Step 5 of Algorithm 1 to select the polynomial terms. For each $\lambda_2$, we calculate the AIC, and choose the $(\lambda_1, \lambda_2)$ with the smallest AIC.

One may wonder whether Step 1 incorrectly discards features required for building a good model in Step 2. We next show theoretically this is not the case. Let $S$ be a subset of $\{1, 2, \ldots, p\}$ and its complement $S^c = \{1, 2, \ldots, p\} \setminus S$. Write the feature matrix $X$ as $X = [X(S), X(S^c)]$. Let response $Y = f(X(S)) + \epsilon$, where $f(\cdot)$ is any function and $\epsilon$ is additive noise. Let $n$ be the number of observations and $s$ the size of $S$. We assume that $X$ is deterministic, $p$ and $s$ are fixed, and $\epsilon_i's$ are i.i.d. and follow the Gaussian distribution with mean 0 and variance $\sigma^2$. Our results also hold for zero mean sub-Gaussian noise with parameter $\sigma^2$. More general results regarding general scaling of $n, p$ and $s$ can also be obtained.

Under the following conditions, we show that Step 1 of SPORE-LASSO, the linear LASSO, selects the relevant features even if the response $Y$ depends on predictors $X(S)$ nonlinearly:

1. The columns $(X_j, j = 1, \ldots, p)$ of $X$ are standardized: $\frac{1}{n}X_j^T X_j = 1$, for all $j$;

2. $\Lambda_{\min}(\frac{1}{n}X(S)^T X(S)) \geq c$ with a constant $c > 0$;

3. $\min |(X(S)^T X(S))^{-1} X(S)^T f(X(S))| > \alpha$ with a constant $\alpha > 0$;

4. $\frac{X_{S^c}^T [I - X_S(X_S^T X_S)^{-1} X_S^T] f(X_S)}{n} < \frac{\eta \alpha c}{2\sqrt{s+1}}$, for some $0 < \eta < 1$;

5. $\|X_{S^c}^T X_S(X_S^T X_S)^{-1}\|_\infty \leq 1 - \eta$;

where $\Lambda_{\min}(\cdot)$ denotes the minimum eigenvalue of a matrix, $\|A\|_\infty$ is defined as $\max_i \left[\sum_j |A_{ij}|\right]$ and the inequalities are defined element-wise.

**Theorem 3.1.** *Under the conditions above, with probability $\to 1$ as $n \to \infty$, there exists some $\lambda$, such that $\hat{\beta} = (\hat{\beta}_S, \hat{\beta}_{S^c})$ is the unique solution of the LASSO (Equation (1)), where $\hat{\beta}_j \neq 0$, for all $j \in S$ and $\hat{\beta}_{S^c} = 0$.*

**Remark.** The first two conditions are trivial: Condition 1 can be obtained by rescaling while Condition 2 assumes that the design matrix composed of the true predictors in the model is not singular. Condition 3 is a reasonable condition which means that the linear projection of the expected response to the space spanned by true predictors is not degenerated. Condition 4 is a little bit tricky; it says that the irrelevant predictors $(X_{S^c})$ are not very correlated with the "residuals" of $E(Y)$ after its projection onto $X_S$. Condition 5 is always needed when considering LASSO's model selection consistency [26, 28]. The proof of the theorem is included in the supplementary material.

### 3.2.2 Adaptive Forward-Backward: SPORE-FoBa

Using *all* of the polynomial expansions of a feature subset is not flexible. In this section, we propose the SPORE-FoBa algorithm, a more flexible algorithm using adaptive forward-backward searching over the polynomially expanded data: during search step $k$ with an active set $T^{(k)}$, we examine one new feature $X_j$, and consider a small candidate set which consists of the candidate feature $X_j$, its higher order terms, and the (non-linear) interactions between previously selected features (indexed by $S$) and candidate feature $X_j$ with total degree up to $d$, i.e., terms with form

$$X_j^{d_1} \Pi_{l \in S} X_l^{d_l}, \quad \text{with } d_1 > 0, d_l \geq 0, \text{ and } d_1 + \sum d_l \leq d. \tag{3}$$

Algorithm 2 below is a short description of the SPORE-FoBa, which uses linear FoBa [27] at step 5and 6. The main idea of SPORE-FoBa is that a term from the candidate set is added into the model if and only if adding this term makes the residual sum of squares ($RSS$) decrease a lot. We scan all of the terms in the candidate set and choose the one which makes the $RSS$ drop most. If the drop in the $RSS$ is greater than a pre-specified value $\epsilon$, we add that term to the active set, which contains the currently selected terms by the SPORE-FoBa algorithm. When considering deleting one term from the active set, we choose the one that makes the sum of residuals increase the least. If this increment is small enough, we delete that term from our current active set.

---
**Algorithm 2** SPORE-FoBa

**Input:** response $Y$, feature columns $X_1, \ldots, X_p$, the maximum degree $d$
**Output:** polynomial terms and the weights
 1: Let $T = \emptyset$
 2: **while** true **do**
 3:     **for** $j = 1, \ldots, p$ **do**
 4:         Let $C$ be the candidate set that contains non-linear and interaction terms from Equation (3)
 5:         Use Linear FoBa to select terms from $C$ to form the new active set $T$.
 6:         Use Linear FoBa to delete terms from $T$ to form a new active set $T$.
 7:     **if** no terms can be added or deleted **then**
 8:         break

---

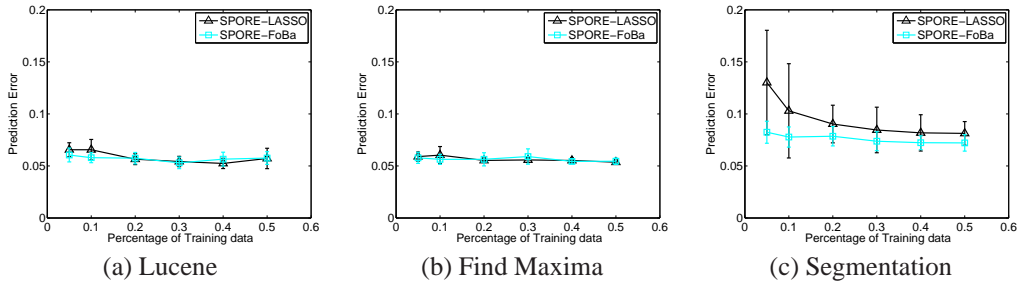

| (a) Lucene | (b) Find Maxima | (c) Segmentation |

Figure 1: Prediction errors of our algorithms across the three data sets varying training-set fractions.

## 4 Evaluation Results

We now experimentally demonstrate that our algorithms are practical, give highly accurate predictors for real problems with small training-set sizes, compare favorably in accuracy to other state-of-the-art sparse-regression algorithms, and produce interpretable, intuitive models.

To evaluate our algorithms, we use as case studies three programs: the Lucene Search Engine [4], and two image processing algorithms, one for finding maxima and one for segmenting an image (both of which are implemented within the ImageJ image processing framework [3]). We chose all three programs according to two criteria. First and most importantly, we sought programs with high variability in the predicted measure (execution time), especially in the face of otherwise similar inputs (e.g., image files of roughly the same size for image processing). Second, we sought programs that implement reasonably complex functionality, for which an inexperienced observer would not be able to trivially identify the important features.

Our collected datasets are as follows. For Lucene, we used a variety of text input queries from two corpora: the works of Shakespeare and the King James Bible. We collected a data set with $n = 3840$ samples, each of which consists of an execution time and a total of $p = 126$ automatically generated features. The time values are in range of $(0.88, 1.13)$ with standard deviation 0.19. For the Find Maxima program within the ImageJ framework, we collected $n = 3045$ samples (from an equal number of distinct, diverse images obtained from three vision corpora [1, 2, 5]), and a total of $p = 182$ features. The execution time values are in range of $(0.09, 2.99)$ with standard deviation 0.24. Finally, from the Segmentation program within the same ImageJ framework on the same image set, we collected again $n = 3045$ samples, and a total of $p = 816$ features for each. The time values are in range of $(0.21, 58.05)$ with standard deviation 3.05. In all the experiments, we fix degree $d = 3$ for polynomial expansion, and normalized each column of feature data into range $[0, 1]$.

**Prediction Error.** We first show that our algorithms predict accurately, even when training on a small number of samples, in both absolute and relative terms. The accuracy measure we use is the relative prediction error defined as $\frac{1}{n_t} \sum |\frac{\hat{y}_i - y_i}{y_i}|$, where $n_t$ is the size of the test data set, and $\hat{y}_i$'s and $y_i$'s are the predicted and actual responses of test data, respectively.

We randomly split every data set into a training set and a test set for a given training-set fraction, train the algorithms and measure their prediction error on the test data. For each training fraction, we repeat the "splitting, training and testing" procedure 10 times and show the mean and standard deviation of prediction error in Figure 1. We see that our algorithms have high prediction accuracy, even when training on only $10\%$ or less of the data (roughly 300 - 400 samples). Specifically, both of our algorithms can achieve less than $7\%$ prediction error on both Lucene and Find Maxima datasets; on the segmentation dataset, SPORE-FoBa achieves less than $8\%$ prediction error, and SPORE-LASSO achieves around $10\%$ prediction error on average.

**Comparisons to State-of-the-Art.** We compare our algorithms to several existing sparse regression methods by examining their prediction errors at different *sparsity* levels (the number of features used in the model), and show our algorithms can clearly outperform LASSO, FoBa and recently proposed non-parametric greedy methods [18] (Figure 2). As a non-parametric greedy algorithm, we use Additive Forward Regression (AFR), because it is faster and often achieves better prediction accuracy than Generalized Forward Regression (GFR) algorithms. We use the Glmnet Matlab implementa-

tion of LASSO and to obtain the LASSO solution path [10]. Since FoBa and SPORE-FoBa naturally produce a path by adding or deleting features (or terms), we record the prediction error at each step. When two steps have the same sparsity level, we report the smallest prediction error. To generate the solution path for SPORE-LASSO, we first use Glmnet to generate a solution path for linear LASSO; then at each sparsity level $k$, we perform full polynomial expansion with $d = 3$ on the selected $k$ features, obtain a solution path on the expanded data, and choose the model with the smallest prediction error among all models computed from all active feature sets of size $k$. From the figure, we see that our SPORE algorithms have comparable performance, and both of them clearly achieve better prediction accuracy than LASSO, FoBa, and AFR. None of the existing methods can build models within 10% of relative prediction error. We believe this is because execution time of a computer program often depends on non-linear combinations of different features, which is usually not well-handled by either linear methods or the additive non-parametric methods. Instead, both of our algorithms can select 2-3 high-quality features and build models with non-linear combinations of them to predict execution time with high accuracy.

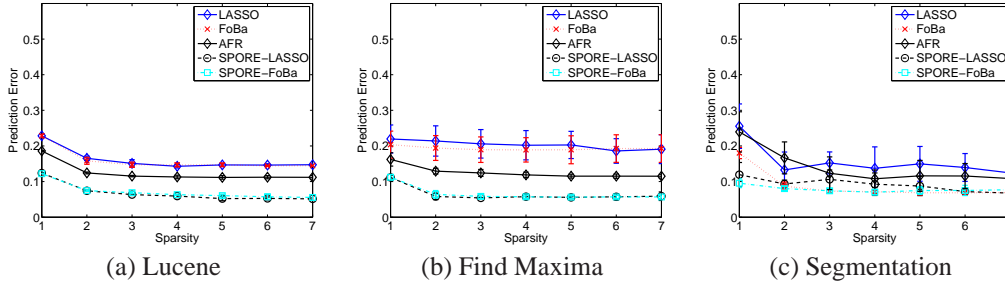

(a) Lucene          (b) Find Maxima          (c) Segmentation

Figure 2: Performance of the algorithms: relative prediction error versus sparsity level.

**Model Interpretability.** To gain better understanding, we investigate the details of the model constructed by SPORE-FoBa for Find Maxima. Our conclusions are similar for the other case studies, but we omit them due to space. We see that with different training set fractions and with different sparsity configurations, SPORE-FoBa can always select two high-quality features from hundreds of automatically generated ones. By consulting with experts of the Find Maxima program, we find that the two selected features correspond to the width ($w$) and height ($h$) of the region of interest in the image, which may in practice differ from the actual image width and height. Those are indeed the most important factors for determining the execution time of the particular algorithm used. For a 10% training set fraction and $\epsilon = 0.01$, SPORE-FoBa obtained

$$f(w, h) = 0.1 + 0.22w + 0.23h + 1.93wh + 0.24wh^2$$

which uses non-linear feature terms(e.g., $wh$, $wh^2$) to predict the execution time accurately (around 5.5% prediction error). Especially when Find Maxima is used as a component of a more complex image processing pipeline, this model would not be the most obvious choice even an expert would pick. On the contrary, as observed in our experiments, neither the linear nor the additive sparse methods handle well such nonlinear terms, and result in inferior prediction performance. A more detailed comparison across different methods is the subject of our on-going work.

## 5 Conclusion

In this paper, we proposed the SPORE (Sparse POlynomial REgression) methodology to build the relationship between execution time of computer programs and features of the programs. We introduced two algorithms to learn a SPORE model, and showed that both algorithms can predict execution time with more than 93% accuracy for the applications we tested. For the three test cases, these results present a significant improvement (a 40% or more reduction in prediction error) over other sparse modeling techniques in the literature when applied to this problem. Hence our work provides one convincing example of using sparse non-linear regression techniques to solve real problems. Moreover, the SPORE methodology is a general methodology that can be used to model computer program performance metrics other than execution time and solve problems from other areas of science and engineering.

# References

[1] Caltech 101 Object Categories. `http://www.vision.caltech.edu/Image_Datasets/Caltech101/Caltech101.html`.

[2] Event Dataset. `http://vision.stanford.edu/lijiali/event_dataset/`.

[3] ImageJ. `http://rsbweb.nih.gov/ij/`.

[4] Mahout. `lucene.apache.org/mahout`.

[5] Visual Object Classes Challenge 2008. `http://pascallin.ecs.soton.ac.uk/challenges/VOC/voc2008/`.

[6] S. Chen, K. Joshi, M. A. Hiltunen, W. H. Sanders, and R. D. Schlichting. Link gradients: Predicting the impact of network latency on multitier applications. In *INFOCOM*, 2009.

[7] B.-G. Chun, L. Huang, S. Lee, P. Maniatis, and M. Naik. Mantis: Predicting system performance through program analysis and modeling. *Technical Report*, 2010. arXiv:1010.0019v1 [cs.PF].

[8] D. Donoho. For most large underdetermined systems of equations, the minimal 1-norm solution is the sparsest solution. *Communications on Pure and Applied Mathematics*, 59:797829, 2006.

[9] B. Efron, T. Hastie, I. Johnstone, and R. Tibshirani. Least angle regression. *Annals of Statistics*, 32(2):407–499, 2002.

[10] J. Friedman, T. Hastie, and R. Tibshirani. Regularization paths for generalized linear models via coordinate descent. *Journal of Statistical Software*, 33(1), 2010.

[11] A. Ganapathi, H. Kuno, U. Dayal, J. L. Wiener, A. Fox, M. Jordan, and D. Patterson. Predicting multiple metrics for queries: Better decisions enabled by machine learning. In *ICDE*, 2009.

[12] S. Goldsmith, A. Aiken, and D. Wilkerson. Measuring empirical computational complexity. In *FSE*, 2007.

[13] C. Gupta, A. Mehta, and U. Dayal. PQR: Predicting query execution times for autonomous workload management. In *ICAC*, 2008.

[14] T. Hastie, R. Tibshirani, and J. Friedman. *The Elements of Statistical Learning*. Springer, 2009.

[15] M. Isard, V. Prabhakaran, J. Currey, U. Wieder, K. Talwar, and A. Goldberg. Quincy: fair scheduling for distributed computing clusters. In *Proceedings of SOSP'09*, 2009.

[16] S.-J. Kim, K. Koh, M. Lustig, S. Boyd, and D. Gorinevsky. An interior-point method for large-scale l1-regularized least squares. *IEEE Journal on Selected Topics in Signal Processing*, 1(4):606–617, 2007.

[17] Z. Li, M. Zhang, Z. Zhu, Y. Chen, A. Greenberg, and Y.-M. Wang. WebProphet: Automating performance prediction for web services. In *NSDI*, 2010.

[18] H. Liu and X. Chen. Nonparametric greedy algorithm for the sparse learning problems. In *NIPS 22*, 2009.

[19] M. Osborne, B. Presnell, and B. Turlach. On the lasso and its dual. *Journal of Computational and Graphical Statistics*, 9(2):319–337, 2000.

[20] P. Ravikumar, J. Lafferty, H. Liu, and L. Wasserman. Sparse additive models. *Journal of the Royal Statistical Society: Series B(Statistical Methodology)*, 71(5):1009–1030, 2009.

[21] P. Ravikumar, V. Vu, B. Yu, T. Naselaris, K. Kay, J. Gallant, and C. Berkeley. Nonparametric sparse hierarchical models describe v1 fmri responses to natural images. *Advances in Neural Information Processing Systems (NIPS)*, 21, 2008.

[22] S. A. Seshia and A. Rakhlin. Game-theoretic timing analysis. In *Proceedings of the IEEE/ACM International Conference on Computer-Aided Design (ICCAD)*, pages 575–582. IEEE Press, Nov. 2008.

[23] S. A. Seshia and A. Rakhlin. Quantitative analysis of systems using game-theoretic learning. *ACM Transactions on Embedded Computing Systems (TECS)*, 2010. To appear.

[24] M. Tariq, A. Zeitoun, V. Valancius, N. Feamster, and M. Ammar. Answering what-if deployment and configuration questions with wise. In *ACM SIGCOMM*, 2008.

[25] R. Tibshirani. Regression shrinkage and selection via the lasso. *J. Royal. Statist. Soc B.*, 1996.

[26] M. Wainwright. Sharp thresholds for high-dimensional and noisy sparsity recovery using l1-constrained quadratic programming (Lasso). *IEEE Trans. Information Theory*, 55:2183–2202, 2009.

[27] T. Zhang. Adaptive forward-backward greedy algorithm for sparse learning with linear models. *Advances in Neural Information Processing Systems*, 22, 2008.

[28] P. Zhao and B. Yu. On model selection consistency of Lasso. *The Journal of Machine Learning Research*, 7:2563, 2006.

[29] H. Zou. The adaptive lasso and its oracle properties. *Journal of the American Statistical Association*, 101(476):1418–1429, 2006.

